# Unifying Non-Maximum Likelihood Learning Objectives with Minimum KL Contraction

**Siwei Lyu**
Computer Science Department
University at Albany, State University of New York
`lsw@cs.albany.edu`

## Abstract

When used to learn high dimensional parametric probabilistic models, the classical maximum likelihood (ML) learning often suffers from computational intractability, which motivates the active developments of non-ML learning methods. Yet, because of their divergent motivations and forms, the objective functions of many non-ML learning methods are seemingly unrelated, and there lacks a unified framework to understand them. In this work, based on an information geometric view of parametric learning, we introduce a general non-ML learning principle termed as *minimum KL contraction*, where we seek optimal parameters that minimizes the contraction of the KL divergence between the two distributions after they are transformed with a *KL contraction operator*. We then show that the objective functions of several important or recently developed non-ML learning methods, including contrastive divergence [12], noise-contrastive estimation [11], partial likelihood [7], non-local contrastive objectives [31], score matching [14], pseudo-likelihood [3], maximum conditional likelihood [17], maximum mutual information [2], maximum marginal likelihood [9], and conditional and marginal composite likelihood [24], can be unified under the minimum KL contraction framework with different choices of the KL contraction operators.

## 1 Introduction

Fitting parametric probabilistic models to data is a basic task in statistics and machine learning. Given a set of training data $\{\mathbf{x}^{(1)}, \cdots, \mathbf{x}^{(n)}\}$, parameter learning aims to find a member in a parametric distribution family, $q_\theta$, to best represent the training data. In practice, many useful high dimensional parametric probabilistic models, such as Markov random fields [18] or products of experts [12], are defined as $q_\theta(\mathbf{x}) = \tilde{q}_\theta(\mathbf{x})/Z(\theta)$, where $\tilde{q}_\theta$ is the unnormalized model and $Z(\theta) = \int_{\mathcal{R}^d} \tilde{q}_\theta(\mathbf{x})d\mathbf{x}$ is the partition function. The maximum (log) likelihood (ML) estimation is the most commonly used method for parameter learning, where the optimal parameter is obtained by solving $\mathrm{argmax}_\theta \frac{1}{n} \sum_{k=1}^n \log q_\theta(\mathbf{x}^{(k)})$. The obtained ML estimators has many desirable properties, such as consistency and asymptotic normality [21]. However, because of the high dimensional integration/summation, the partition function in $q_\theta$ oftentimes makes ML learning computationally intractable. For this reason, non-ML parameter learning methods that use "tricks" to obviate direct computation of the partition function have experienced rapid developments, particularly in recent years. While many computationally efficient non-ML learning methods have achieved impressive practical performances, with a few exceptions, their different learning objectives and numerical implementations seem to suggest that they are largely unrelated.

In this work, based on the information geometric view of parametric learning, we elaborate on a general non-ML learning principle termed as *minimum KL contraction* (MKC), where we seek optimal parameters that minimize the contraction of the KL divergence between two distributions after they are transformed with a *KL contraction operator*. The KL contraction operator is a mapping between

probability distributions under which the KL divergence of two distributions tend to reduce unless they are equal. We then show that the objective functions of a wide range of non-ML learning methods, including contrastive divergence [12], noise-contrastive estimation [11], partial likelihood [7], non-local contrastive objectives [31], score matching [14], pseudo-likelihood [3], maximum conditional likelihood [17], maximum mutual information [2], maximum marginal likelihood [9], and conditional and marginal composite likelihood [24], can all be unified under the MKC framework with different choices of the KL contraction operators and MKC objective functions.

## 2 Related Works

Similarities in the parameter updates among non-ML learning methods have been noticed in several recent works. For instance, in [15], it is shown that the parameter update in score matching [14] is equivalent to the parameter update in a version of contrastive divergence [12] that performs Langevin approximation instead of Gibbs sampling, and both are approximations to the parameter update of pseudo-likelihood [3]. This connection is further generalized in [1], which shows that parameter update in another variant of contrastive divergence is equivalent to a stochastic parameter update in conditional composite likelihood [24]. However, such similarities in numerical implementations are only tangential to the more fundamental relationship among the *objective functions* of different non-ML learning methods. On the other hand, the *energy based learning* [22] presents a general framework that subsume most non-ML learning objectives, but its broad generality also obscures their specific statistical interpretations.

At the objective function level, relations between some non-ML methods are known. For instance, it is known that pseudo-likelihood is a special case of conditional composite likelihood [30]. In [10], several non-ML learning methods are unified under the framework of minimizing Bregman divergence.

## 3 KL Contraction Operator

We base our discussion hereafter on continuous variables and probability density functions. Most results can be readily extended to the discrete case by replacing integrations and probability density functions with summations and probability mass functions. We denote $\Omega_d$ as the set of all probability density functions over $\mathcal{R}^d$. For two different probability distributions $p, q \in \Omega_d$, their Kulback-Leibler (KL) divergence (also known as relative entropy or I-divergence) [6] is defined as $\mathrm{KL}(p\|q) = \int_{\mathcal{R}^d} p(\mathbf{x}) \log \frac{p(\mathbf{x})}{q(\mathbf{x})} d\mathbf{x}$. KL divergence is non-negative and equals to zero if and only if $p = q$ *almost everywhere* (*a.e.*). We define a distribution operator, $\Phi$, as a mapping between a density function $p \in \Omega_d$ to another density function $\tilde{p} \in \Omega_{d'}$, and adopt the shorthand notation $\tilde{p} = \Phi\{p\}$. A distribution $p$ is a *fix point* of a distribution operator $\Phi$ if $p = \Phi\{p\}$.

A KL contraction operator is a distribution operator, $\Phi : \Omega_d \mapsto \Omega_{d'}$, such that for any $p, q \in \Omega_d$, there exist a constant $\beta \geq 1$ for the following condition to hold:

$$\mathrm{KL}(p\|q) - \beta \mathrm{KL}(\Phi\{p\}\|\Phi\{q\}) \geq 0. \qquad (1)$$

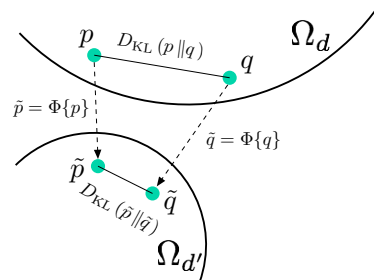

Figure 1: *Illustration of a KL contraction operator on two density functions $p$ and $q$.*

Subsequently, $\beta$ is known as the *contraction factor*, and LHS of Eq.(1) is the *KL contraction* of $p$ and $q$ under $\Phi$. Obviously, if $p = q$ (*a.e.*), their KL contraction, as well as their KL divergence, is zero. In addition, a KL contraction operator is *strict* if the equality in Eq.(1) holds only when $p = q$ (*a.e.*). Intuitively, if the KL divergence is regarded as a "distance" metric of probability distributions[1], then it is never increased after both distributions are transformed with a KL contraction operator, a graphical illustration of which is shown in Fig.1. Furthermore, under a strict KL contraction operator, the KL divergence is always reduced unless the two distributions are equal (*a.e.*). The KL contraction operators are analogous to the contraction operators in ordinary metric spaces, with $\beta$ having a similar role as the Lipschitz constant [19].

Eq.(1) gives the general and abstract definition of KL contraction operators. In the following, we give several examples of KL contraction operators that are constructed from common operations of probability distributions.

## 3.1 Conditional Distribution

We can form a family of KL contraction operators using conditional distributions. Consider $\mathbf{x} \in \mathcal{R}^d$ with distribution $p(\mathbf{x}) \in \Omega_d$ and $\mathbf{y} \in \mathcal{R}^{d'}$, from a conditional distribution, $T(\mathbf{y}|\mathbf{x})$, we can define a distribution operator, as

$$\Phi_T^c\{p\}(\mathbf{y}) = \int_{\mathcal{R}^d} T(\mathbf{y}|\mathbf{x})p(\mathbf{x})d\mathbf{x} = \tilde{p}(\mathbf{y}). \tag{2}$$

The following result shows that $\Phi_T^c$ is a strict KL contraction operator with $\beta = 1$.

**Lemma 1 (Cover & Thomas [6][2])** *For two distributions $p, q \in \Omega_d$, with the distribution operator defined in Eq.(2), we have*

$$\mathrm{KL}(p\|q) - \mathrm{KL}(\Phi_T^c\{p\}\|\Phi_T^c\{q\}) = \int_{\mathcal{R}^{d'}} \tilde{p}(\mathbf{y})\mathrm{KL}(T_p(\mathbf{x}|\mathbf{y})\|T_q(\mathbf{x}|\mathbf{y}))\, d\mathbf{y} \geq 0,$$

*where $T_p(\mathbf{x}|\mathbf{y}) = \frac{T(\mathbf{y}|\mathbf{x})p(\mathbf{x})}{\tilde{p}(\mathbf{y})}$ and $T_q(\mathbf{x}|\mathbf{y}) = \frac{T(\mathbf{y}|\mathbf{x})q(\mathbf{x})}{\tilde{q}(\mathbf{y})}$ are the induced conditional distributions from $p$ and $q$ with $T$. Furthermore, the equality holds if and only if $p = q$ (a.e.).*

## 3.2 Marginalization and Marginal Grafting

Two related types of KL contraction operators can be constructed based on marginal distributions. Consider $\mathbf{x}$ with distribution $p(\mathbf{x}) \in \Omega_d$, and a nonempty index subset $A \subset \{1, \cdots, d\}$. Let $\backslash A$ denote $\{1, \cdots, d\} - A$, the marginal distribution, $p_A(\mathbf{x}_A)$, of sub-vector $\mathbf{x}_A$ formed by components whose indices are in $A$ is obtained by integrating $p(\mathbf{x})$ over sub-vector $\mathbf{x}_{\backslash A}$. This marginalization operation thus defines a distribution operator between $p \in \Omega_d$ and $p_A \in \Omega_{|A|}$, as:

$$\Phi_A^m\{p\}(\mathbf{x}_A) = \int_{\mathcal{R}^{d-|A|}} p(\mathbf{x})d\mathbf{x}_{\backslash A} = p_A(\mathbf{x}_A) \tag{3}$$

Another KL contraction operator termed as *marginal grafting* can also be defined based on $p_A$. For a distribution $q(\mathbf{x}) \in \Omega_d$, the marginal grafting operator is defined as:

$$\Phi_{p,A}^g\{q\}(\mathbf{x}) = \frac{q(\mathbf{x})p_A(\mathbf{x}_A)}{q_A(\mathbf{x}_A)} = q_{\backslash A|A}(\mathbf{x}_{\backslash A}|\mathbf{x}_A)p_A(\mathbf{x}_A), \tag{4}$$

$\Phi_{p,A}^g\{q\}$ can be understood as replacing $q_A$ in $q(\mathbf{x})$ with $p_A$. The last term in Eq.4 is nonnegative and integrates to one over $\mathcal{R}^d$, and is thus a proper probability distribution in $\Omega_d$. Furthermore, $p$ is the fixed point of operator $\Phi_{p,A}^g$, as $\Phi_{p,A}^g\{p\} = p$.

The following result shows that both $\Phi_A^m$ and $\Phi_{p,A}^{mg}$ are KL contraction operators, and that they are in a sense complementary to each other.

**Lemma 2 (Huber [13])** *For two distributions $p, q \in \Omega_d$, with the distribution operators defined in Eq.(3) and Eq.(4), we have*

$$\mathrm{KL}(p\|q) - \mathrm{KL}\left(\Phi_{p,A}^g\{p\} \middle\| \Phi_{p,A}^g\{q\}\right) = \mathrm{KL}(\Phi_A^m\{p\}\|\Phi_A^m\{q\}) .$$

*Furthermore,*

$$\mathrm{KL}\left(\Phi_{p,A}^g\{p\} \middle\| \Phi_{p,A}^g\{q\}\right) = \int_{\mathcal{R}^d} p_A(\mathbf{x}_A)\mathrm{KL}\left(p_{\backslash A|A}(\cdot|\mathbf{x}_A)\|q_{\backslash A|A}(\cdot|\mathbf{x}_A)\right) d\mathbf{x}_A,$$

*where $p_{\backslash A|A}(\cdot|\mathbf{x}_A)$ and $q_{\backslash A|A}(\cdot|\mathbf{x}_A)$ are the conditional distributions induced from $p(\mathbf{x})$ and $q(\mathbf{x})$, and*

$$\mathrm{KL}(\Phi_A^m\{p\}\|\Phi_A^m\{q\}) = \mathrm{KL}(p_A(\mathbf{x}_A)\|q_A(\mathbf{x}_A)) .$$

Lemma 2 also indicates that neither $\Phi_A^m$ nor $\Phi_{p,A}^{mg}$ is strict - the KL contraction of $p(\mathbf{x})$ and $q(\mathbf{x})$ for the former is zero if $p_{\backslash A|A}(\mathbf{x}_{\backslash A}|\mathbf{x}_A) = q_{\backslash A|A}(\mathbf{x}_{\backslash A}|\mathbf{x}_A)$ (a.e.), even though they may differ on the marginal distribution over $\mathbf{x}_A$. And for the latter, having $p_A(\mathbf{x}_A) = q_A(\mathbf{x}_A)$ is sufficient to make their KL contraction zero.

### 3.3 Binary Mixture

For two different distributions $p(\mathbf{x})$ and $g(\mathbf{x}) \in \Omega_d$, we introduce a binary variable $c \in \{0, 1\}$ and $Pr(c = i) = \pi_i$, with $\pi_0, \pi_1 \in [0, 1]$ and $\pi_0 + \pi_1 = 1$. We can then form a joint distribution $\hat{p}(\mathbf{x}, c = 0) = \pi_0 g(\mathbf{x})$ and $\hat{p}(\mathbf{x}, c = 1) = \pi_1 p(\mathbf{x})$ over $\mathcal{R}^d \times \{0, 1\}$. Marginalizing out $c$ from $\hat{p}(\mathbf{x}, c)$, we obtain a binary mixture $\tilde{p}(\mathbf{x})$, which induces a distribution operator:

$$\Phi_g^b\{p\}(\mathbf{x}) = \pi_0 g(\mathbf{x}) + \pi_1 p(\mathbf{x}) = \tilde{p}(\mathbf{x}). \tag{5}$$

The following result shows that $\Phi_g^b$ is a strict KL contraction operator with $\beta = 1/\pi_1$.

**Lemma 3** *For two distributions $p, q \in \Omega_d$, with the distribution operator defined in Eq.(5), we have*

$$\text{KL}(p\|q) - \frac{1}{\pi_1}\text{KL}\big(\Phi_g^b\{p\}\big\|\Phi_g^b\{q\}\big) = \frac{1}{\pi_1} \int_{\mathcal{R}^d} \tilde{p}(\mathbf{x}) \left[\text{KL}\big(p_{c|\mathbf{x}}(c|\mathbf{x})\big\|q_{c|\mathbf{x}}(c|\mathbf{x})\big)\right] d\mathbf{x} \geq 0,$$

*where $p(c|\mathbf{x})$ and $q(c|\mathbf{x})$ are the induced posterior conditional distributions from $\hat{p}(\mathbf{x}, c)$ and $\hat{q}(\mathbf{x}, c)$, respectively. The equality holds if and only if $p = q$ (a.e.).*

### 3.4 Lumping

Let $\mathcal{S} = (S_1, S_2, \cdots, S_m)$ be a partition of $\mathcal{R}^d$ such that $S_i \cap S_j = \emptyset$ for $i \neq j$, and $\bigcup_{i=1}^m S_i = \mathcal{R}^d$, the lumping [8] of a distribution $p(\mathbf{x}) \in \Omega_d$ over $\mathcal{S}$ yields a distribution over $i \in \{1, 2, \cdots, m\}$, and subsequently induces a distribution operator $\Phi_{\mathcal{S}}^l$, as:

$$\Phi_{\mathcal{S}}^l\{p\}(i) = \int_{\mathbf{x} \in S_i} p(\mathbf{x}) d\mathbf{x} = P_i^{\mathcal{S}}, \text{ for } i = 1, \cdots, m. \tag{6}$$

The following result shows that $\Phi_{\mathcal{S}}^l$ is a KL contraction operator with $\beta = 1$.

**Lemma 4 (Csiszàr & Shields [8])** *For two distributions $p, q \in \Omega_d$, with the distribution operator defined in Eq.(6), we have*

$$\text{KL}(p\|q) - \text{KL}\big(\Phi_{\mathcal{S}}^l\{p\}\big\|\Phi_{\mathcal{S}}^l\{p\}\big) = \sum_{i=1}^m P_i^{\mathcal{S}}\text{KL}(\tilde{p}_i\|\tilde{q}_i) \geq 0,$$

*where $\tilde{p}_i(\mathbf{x}) = \frac{p(\mathbf{x}) \times \mathbf{1}_{[\mathbf{x} \in S_i]}}{\int_{\mathbf{x}' \in S_i} p(\mathbf{x}')d\mathbf{x}'}$ and $\tilde{q}_i(\mathbf{x}) = \frac{q(\mathbf{x}) \times \mathbf{1}_{[\mathbf{x} \in S_i]}}{\int_{\mathbf{x}' \in S_i} q(\mathbf{x}')d\mathbf{x}'}$ are the distributions induced from $p$ and $q$ by restricting to $S_i$, respectively, with $\mathbf{1}_{[\cdot]}$ being the indicator function.*

Note that $\Phi_{\mathcal{S}}^l$ is in general not strict, as two distributions agree over all $\tilde{p}_i$ and $\tilde{q}_i$ will have a zero KL contraction.

## 4 Minimizing KL Contraction for Parametric Learning

In this work, we take the information geometric view of parameter learning - assuming training data are samples from a distribution $p \in \Omega_d$, we seek an optimal distribution on the statistical manifold corresponding to the parametric distribution family $q_\theta$ that best approximates $p$ [20]. In this context, the maximum (log) likelihood learning is equivalent to finding parameter $\theta$ that minimizes the KL divergence of $p$ and $q_\theta$ [8], as $\text{argmin}_\theta \text{KL}(p\|q_\theta) = \text{argmax}_\theta \int_{\mathcal{R}^d} p(\mathbf{x}) \log q_\theta(\mathbf{x}) d\mathbf{x}$. The data based ML objective is obtained when we approximate the expectation with sample average as $\int_{\mathcal{R}^d} p(\mathbf{x}) \log q_\theta(\mathbf{x}) d\mathbf{x} \approx \frac{1}{n} \sum_{k=1}^n \log q_\theta(\mathbf{x}^{(k)})$.

The KL contraction operators suggest an alternative approach for parametric learning. In particular, the KL contraction of $p$ and $q_\theta$ under a KL contraction operator is always nonnegative and reaches zero when $p$ and $q_\theta$ are equal almost everywhere. Therefore, we can minimize their KL contraction under a KL contraction operator to encourage the matching of $q_\theta$ to $p$. We term this general approach of parameter learning as *minimum KL contraction* (MKC). Mathematically, minimum KL contraction may be realized with three different but related types of objective functions.

- **Type I**: With a KL contraction operator $\Phi$, we can find optimal $\theta$ that directly minimizes the KL contraction between $p$ and $q_\theta$, as:

$$\underset{\theta}{\text{argmin}}\, \text{KL}(p\|q_\theta) - \beta\text{KL}(\Phi\{p\}\|\Phi\{q_\theta\}). \tag{7}$$

In practice, it may be desirable to use $\Phi$ with $\beta = 1$ that is also a linear operator for $L_2$ bounded functions over $\mathcal{R}^d$ [19]. To better see this, consider $q_\theta(\mathbf{x}) = \frac{\tilde{q}_\theta(\mathbf{x})}{Z(\theta)}$ as the model defined with

the unnormalized model and its partition function. Furthermore, assuming that we can obtain samples $\{\mathbf{x}_1, \cdots, \mathbf{x}_n\}$ and $\{\mathbf{y}_1, \cdots, \mathbf{y}_{n'}\}$ from $p$ and $\Phi\{p\}$, respectively, the optimization of Eq.(7) can be approximated as

$$\underset{\theta}{\operatorname{argmin}}\, \mathrm{KL}(p\|q_\theta) - \mathrm{KL}(\Phi\{p\}\|\Phi\{q_\theta\}) \approx \underset{\theta}{\operatorname{argmax}}\, \frac{1}{n}\sum_{k=1}^{n}\log\tilde{q}_\theta(\mathbf{x}^{(k)}) - \frac{1}{n'}\sum_{k=1}^{n'}\log\Phi\{\tilde{q}_\theta\}(\mathbf{y}^{(k)}),$$

where due to the linearity of $\Phi$, the two terms of $Z(\overline{\theta})^{-1}$ in $q_\theta$ and $L\{q_\theta\}$ cancel out each other. Therefore, the optimization does not require the computation of the partition function, a highly desirable property for fitting parameters in high dimensional probabilistic models with intractable partition functions. Type I MKC objective functions with KL contraction operators induced from conditional distribution, marginalization, marginal grafting, linear transform, and lumping all fall into this category. However, using nonlinear KL contraction operators, such as the one induced from binary mixtures, may also be able to avoid computing the partition function (e.g., Section 4.4). Furthermore, the KL contraction operator in Eq.(7) can have parameters, which can include the model parameter $\theta$ (e.g., Section 4.2). However, the optimization becomes more complicated as $\Phi\{p\}$ cannot be ignored when optimizing $\theta$. Last, note that when using Type I MKC objective functions with a non-strict KL contraction operator, we cannot guarantee $p = q_\theta$ even if their corresponding KL contraction is zero.

- **Type II**: Consider a strict KL contraction operator with $\beta = 1$, denoted as $\Phi_t$, is parameterized by an auxiliary parameter $t$ that is different from $\theta$, and for any distribution $p \in \Omega_d$, we have $\Phi_0\{p\} = p$ and $\Phi_t\{p\}$ is continuously differentiable with regards to $t$. Then, the KL divergence $\Phi_t\{p\}$ and $\Phi_t\{q_\theta\}$ can be regarded as a function of $t$ and $\theta$, as: $\mathcal{L}(t,\theta) = \mathrm{KL}(\Phi_t\{p\}\|\Phi_t\{q_\theta\})$. Thus, the KL contraction in Eq.(7) can be approximated with a Taylor expansion near $t = 0$, as $\mathrm{KL}(p\|q_\theta) - \mathrm{KL}(\Phi_{\delta t}\{p\}\|\Phi_{\delta t}\{q_\theta\}) = \mathrm{KL}(\Phi_0\{p\}\|\Phi_0\{q_\theta\}) - \mathrm{KL}(\Phi_{\delta t}\{p\}\|\Phi_{\delta t}\{q_\theta\}) = \mathcal{L}(0,\theta) - \mathcal{L}(\delta t,\theta) \approx -\delta t\, \frac{\partial\mathcal{L}(t,\theta)}{\partial t}\Big|_{t=0} = -\delta t\, \frac{\partial}{\partial t}\mathrm{KL}(\Phi_t\{p\}\|\Phi_t\{q_\theta\})\big|_{t=0}$. If the derivative of KL contraction with regards to $t$ is easier to work with than the KL contraction itself (e.g., Section 4.5), we can fix $\delta t$ and equivalently maximizing the derivative, which is the Type II MKC objective function, as

$$\underset{\theta}{\operatorname{argmax}}\, \frac{\partial}{\partial t}\mathrm{KL}(\Phi_t\{p\}\|\Phi_t\{q_\theta\})\Big|_{t=0}. \tag{8}$$

- **Type III**: In the case where we have access to a set of different KL contraction operators, $\{\Phi_1, \cdots, \Phi_m\}$, we can implement the minimum KL contraction principle by finding optimal $\theta$ that minimizes their average KL contraction, as

$$\underset{\theta}{\operatorname{argmin}}\, \frac{1}{m}\sum_{i=1}^{m}\left(\mathrm{KL}(p\|q_\theta) - \beta_i\mathrm{KL}(\Phi_i\{p\}\|\Phi_i\{q_\theta\})\right). \tag{9}$$

As each KL contraction in the sum is nonnegative, Eq.(9) is zero if and only if each KL contraction is zero. If the consistency of $p$ and $q_\theta$ with regards to $\Phi_i$ corresponds to certain constraints on $q_\theta$, the objective function, Eq.(9), represents the consistency of all such constraints. Under some special cases, minimizing Eq.(9) to zero over a sufficient number of certain types of KL contraction operators may indeed ensure equality of $p$ and $q_\theta$ (e.g., Section 4.6).

## 4.1 Fitting Gaussian Model with KL Contraction Operator from a Gaussian Distribution

We first describe an instance of MKC learning under a very simple setting, where we approximate a distribution $p(x)$ for $x \in \mathcal{R}$ with known mean $\mu_0$ and variance $\sigma_0^2$, with a Gaussian model $q_\theta$ whose mean and variance are the parameters to be estimated as $\theta = (\mu, \sigma^2)$. Using the strict KL contraction operator $\Phi_T^c$ constructed with a Gaussian conditional distribution

$$T(y|x) = \frac{1}{\sqrt{2\pi\sigma_1^2}}\exp\left(-\frac{(y-x)^2}{2\sigma_1^2}\right),$$

with known variance $\sigma_1^2$, we form the Type I MKC objective function. In this simple case, Eq.(7) is reduced to a closed form objective function, as:

$$\underset{\mu,\sigma^2}{\operatorname{argmin}}\left[\frac{\sigma_0^2}{2\sigma^2} - \frac{\sigma_0^2 + \sigma_1^2}{2(\sigma^2 + \sigma_1^2)} + \frac{1}{2}\log\frac{\sigma^2}{\sigma^2 + \sigma_1^2} + \frac{\sigma_1^2(\mu - \mu_0)^2}{2\sigma^2(\sigma^2 + \sigma_1^2)}\right],$$

whose optimal solution, $\mu = \mu_0$ and $\sigma^2 = \sigma_0^2$, is obtained by direct differentiation. The detailed derivation of this result is omitted due to the limit of space. Note that, the optimal parameters do not rely on the parameter in the KL contraction operator (in this case, $\sigma_1^2$), and are the same as those obtained by minimizing the KL divergence between $p$ and $q_\theta$, or equivalently, maximizing the log likelihood, when samples from $p(x)$ are used to approximate the expectation.

## 4.2 Relation with Contrastive Divergence [12]

Next, we consider the general strict KL contraction operator $\Phi^c_{T_\theta}$ constructed from a conditional distribution, $T_\theta(\mathbf{y}|\mathbf{x})$, for $\mathbf{x}, \mathbf{y} \in \mathcal{R}^d$, of which the parametric model $q_\theta$ is a fixed point, as $q_\theta(\mathbf{y}) = \int_{\mathcal{R}^d} T_\theta(\mathbf{y}|\mathbf{x}) q_\theta(\mathbf{x}) d\mathbf{x} = \Phi^c_{T_\theta}\{q_\theta\}(\mathbf{y})$. In other words, $q_\theta$ is the *equilibrium distribution* of the Markov chain whose transitional distribution is given by $T_\theta(\mathbf{y}|\mathbf{x})$. The Type I objective function of minimum KL contraction, Eq.(7), for $p, q_\theta \in \Omega_d$ under $\Phi^c_{T_\theta}$ is

$$\underset{\theta}{\operatorname{argmin}} \, \mathrm{KL}(p\|q_\theta) - \mathrm{KL}\big(\Phi^c_{T_\theta}\{p\}\big\|\Phi^c_{T_\theta}\{q_\theta\}\big) = \underset{\theta}{\operatorname{argmin}} \, \mathrm{KL}(p\|q_\theta) - \mathrm{KL}(p_\theta\|q_\theta),$$

where $p_\theta$ is the shorthand notation for $\Phi^c_{T_\theta}\{p\}$. Note that this is the objective function of the contrastive divergence learning [12]. However, the dependency of $p_\theta$ on $\theta$ makes this objective function difficult to optimize. By ignoring this dependency, the practical parameter update in contrastive divergence only approximately follows the gradient of this objective function [5].

## 4.3 Relation with Partial Likelihood [7] and Non-local Contrastive Objectives [31]

Next, we consider the Type I MKC objective function, Eq.(7), combined with the KL contraction operator constructed from lumping. Using Lemma 4, we have

$$\underset{\theta}{\operatorname{argmin}} \big\{ \mathrm{KL}(p\|q_\theta) - \mathrm{KL}\big(\Phi^l_{\mathcal{S}}\{p\}\big\|\Phi^l_{\mathcal{S}}\{q_\theta\}\big) \big\} = \underset{\theta}{\operatorname{argmin}} \sum_{i=1}^m P^{\mathcal{S}}_i \mathrm{KL}\big(\tilde{p}_i\big\|\tilde{q}^\theta_i\big)$$

$$= \underset{\theta}{\operatorname{argmax}} \sum_{i=1}^m P^{\mathcal{S}}_i \int_{\mathbf{x} \in S_i} \tilde{p}_i(\mathbf{x}) \log \tilde{q}^\theta_i(\mathbf{x}) d\mathbf{x} \approx \underset{\theta}{\operatorname{argmax}} \frac{1}{n} \sum_{k=1}^n \mathbf{1}_{\left[\mathbf{x}^{(k)} \in S_i\right]} \sum_{i=1}^m P^{\mathcal{S}}_i \log \tilde{q}^\theta_i(\mathbf{x}^{(k)}),$$

where $\{\mathbf{x}^{(1)}, \cdots, \mathbf{x}^{(n)}\}$ are samples from $p(\mathbf{x})$. Minimizing KL contraction in this case is equivalent to maximizing the weighted sum of log likelihood of the probability distributions formed by restricting the overall model to subsets of state space. The last step resembles the *partial likelihood* objective function [7], which is recently rediscovered in the context of discriminative learning as *non-local contrastive objectives* [31]. In [31], the partitions are required to overlap with each other, while the above result shows that non-overlapping partitions of $\mathcal{R}^d$ can also be used for non-ML parameter learning.

## 4.4 Relation with Noise Contrastive Estimation [11]

Next, we consider the Type I MKC objective function, Eq.(7), combined with the strict KL contraction operator constructed from the binary mixture operation (Lemma 3). In particular, we simplify Eq.(7) using the definition of $\Phi^b_g$, as:

$$\underset{\theta}{\operatorname{argmin}} \, \mathrm{KL}(p\|q_\theta) - \frac{1}{\pi_1}\mathrm{KL}\big(\Phi^b_g\{p\}\big\|\Phi^b_g\{q_\theta\}\big)$$

$$= \underset{\theta}{\operatorname{argmin}} \frac{1}{\pi_1} \int_{\mathcal{R}^d} (\pi_0 g(\mathbf{x}) + \pi_1 p(\mathbf{x})) \log (\pi_0 g(\mathbf{x}) + \pi_1 q_\theta(\mathbf{x})) \, d\mathbf{x} - \int_{\mathcal{R}^d} p(\mathbf{x}) \log q_\theta(\mathbf{x}) d\mathbf{x}$$

$$= \underset{\theta}{\operatorname{argmax}} \int_{\mathcal{R}^d} p(\mathbf{x}) \log \frac{\pi_1 q_\theta(\mathbf{x})}{\pi_0 g(\mathbf{x}) + \pi_1 q_\theta(\mathbf{x})} d\mathbf{x} + \frac{\pi_0}{\pi_1} \int_{\mathcal{R}^d} g(\mathbf{x}) \log \frac{\pi_0 g(\mathbf{x})}{\pi_0 g(\mathbf{x}) + \pi_1 q_\theta(\mathbf{x})} d\mathbf{x}.$$

When the expectations in the above objective function are approximated with averages over samples from $p(\mathbf{x})$ and $g(\mathbf{x})$, $\{\mathbf{x}^{(1)}, \cdots, \mathbf{x}^{(n_+)}\}$ and $\{\mathbf{y}^{(1)}, \cdots, \mathbf{y}^{(n_-)}\}$, the Type I MKC objective function in this case reduces to

$$\underset{\theta}{\operatorname{argmax}} \frac{1}{n_+} \sum_{k=1}^{n_+} \log \frac{\pi_1 q_\theta(\mathbf{x}^{(k)})}{\pi_0 g(\mathbf{x}^{(k)}) + \pi_1 q_\theta(\mathbf{x}^{(k)})} + \frac{\pi_0}{\pi_1} \frac{1}{n_-} \sum_{k=1}^{n_-} \log \frac{\pi_0 g(\mathbf{y}^{(k)})}{\pi_0 g(\mathbf{y}^{(k)}) + \pi_1 q_\theta(\mathbf{y}^{(k)})}.$$

If we set $\pi_0 = \pi_1 = 1/2$, and treat $\{\mathbf{x}^{(1)}, \cdots, \mathbf{x}^{(n_+)}\}$ and $\{\mathbf{y}^{(1)}, \cdots, \mathbf{y}^{(n_-)}\}$ as data of interest and noise, respectively, the above objective function can also be interpreted as minimizing the Bayesian classification error of data and noise, which is the objective function of *noise-contrastive* estimation [11].

## 4.5 Relation with Score Matching [14]

Next, we consider the strict KL contraction operator, $\Phi^c_{T_t}$, constructed from an isotropic Gaussian conditional distribution with a time decaying variance (i.e., a Gaussian diffusion process):

$$T_t(\mathbf{y}|\mathbf{x}) = \frac{1}{(2\pi t)^{d/2}} \exp\left(-\frac{\|\mathbf{y}-\mathbf{x}\|^2}{2t}\right),$$

where $t \in [0, \infty)$ is the continuous temporal index. Note that we have $\Phi^c_{T_0}\{p\} = p$ for any $p \in \Omega_d$. If both $p(\mathbf{x})$ and $q_\theta(\mathbf{x})$ are functions differentiable with regards to $\mathbf{x}$, it is know that the temporal derivative of the KL contraction of $p$ and $q_\theta$ under $\Phi^c_{T_t}$ is in closed form, which is formally stated in the following result.

**Lemma 5 (Lyu [25])** *For any two distributions $p, q \in \Omega_d$ differentiable with regards to $\mathbf{x}$, we have*

$$\frac{d}{dt}\text{KL}\left(\Phi^c_{T_t}\{p\}\big\|\Phi^c_{T_t}\{q_\theta\}\right) = -\frac{1}{2}\int_{\mathcal{R}^d} \Phi^c_{T_t}\{p\}(\mathbf{x}) \left\|\frac{\nabla_{\mathbf{x}}\Phi^c_{T_t}p(\mathbf{x})}{\Phi^c_{T_t}p(\mathbf{x})} - \frac{\nabla_{\mathbf{x}}\Phi^c_{T_t}q_\theta(\mathbf{x})}{\Phi^c_{T_t}q_\theta(\mathbf{x})}\right\|^2 d\mathbf{x}, \quad (10)$$

*where $\nabla_{\mathbf{x}}$ is the gradient operator with regards to $\mathbf{x}$.*

Setting $t = 0$ in Eq.(10), we obtain a closed form for the Type II MKC objective function, Eq.(8), which can be further simplified [14], as

$$\operatorname*{argmax}_\theta \frac{d}{dt}\text{KL}(\Phi_t\{p\}\|\Phi_t\{q_\theta\})\bigg|_{t=0} = \operatorname*{argmin}_\theta \int_{\mathcal{R}^d} p(\mathbf{x}) \left\|\frac{\nabla_{\mathbf{x}}p(\mathbf{x})}{p(\mathbf{x})} - \frac{\nabla_{\mathbf{x}}q_\theta(\mathbf{x})}{q_\theta(\mathbf{x})}\right\|^2 d\mathbf{x}$$

$$= \operatorname*{argmin}_\theta \int_{\mathcal{R}^d} p(\mathbf{x}) \left(\|\nabla_{\mathbf{x}}\log q_\theta(\mathbf{x})\|^2 + 2\triangle_{\mathbf{x}}\log q_\theta(\mathbf{x})\right) d\mathbf{x}$$

$$\approx \operatorname*{argmin}_\theta \frac{1}{n}\sum_{k=1}^{n}\left(\left\|\nabla_{\mathbf{x}}\log q_\theta(\mathbf{x}^{(k)})\right\|^2 + 2\triangle_{\mathbf{x}}\log q_\theta(\mathbf{x}^{(k)})\right),$$

where $\{\mathbf{x}^{(1)}, \cdots, \mathbf{x}^{(n)}\}$ are samples from $p(\mathbf{x})$, and $\triangle_{\mathbf{x}}$ is the Laplacian operator with regards to $\mathbf{x}$. The last step is the objective function of *score matching* learning [14].

## 4.6 Relation with Conditional Composite Likelihood [24] and Pseudo-Likelihood [3]

Next, we consider the Type I MKC objective function, Eq.(7), combined with the KL contraction operator, $\Phi^m_A$, constructed from marginalization. According to Lemma 2, we have $\operatorname{argmin}_\theta \text{KL}(p\|q) - \text{KL}(\Phi^m_A\{p\}\|\Phi^m_A\{q\}) = \operatorname{argmax}_\theta \int_{\mathcal{R}^d} p(\mathbf{x})\log q_{\backslash A|A}(\mathbf{x}_{\backslash A}|\mathbf{x}_A)d\mathbf{x} \approx \operatorname{argmax}_\theta \frac{1}{n}\sum_{k=1}^{n}\log q_{\backslash A|A}(\mathbf{x}^{(k)}_{\backslash A}|\mathbf{x}^{(k)}_A)$, where in the last step, expectation over $p(\mathbf{x})$ is replaced with averages over samples from $p(\mathbf{x})$, $\{\mathbf{x}^{(1)}, \cdots, \mathbf{x}^{(n)}\}$. This corresponds to the objective function in *maximum conditional likelihood* [17] or *maximum mutual information* [2], which are non-ML learning objectives for discriminative learning of high dimensional probabilistic data models.

However, Lemma 2 also shows that $\text{KL}(p\|q) - \text{KL}(\Phi^m_A\{p\}\|\Phi^m_A\{q\}) = 0$ is not sufficient to guarantee $p = q_\theta$. Alternatively, we can use the Type III MKC objective function, Eq.(9), to combine KL contraction operators formed from marginalizations over $m$ different index subsets $A_1, \cdots, A_m$:

$$\operatorname*{argmin}_\theta \text{KL}(p\|q) - \frac{1}{m}\sum_{i=1}^{m}\text{KL}\left(\Phi^m_{A_i}\{p\}\big\|\Phi^m_{A_i}\{q\}\right) \approx \operatorname*{argmax}_\theta \frac{1}{m}\sum_{i=1}^{m}\frac{1}{n}\sum_{k=1}^{n}\log q_{A_i|\backslash A_i}(\mathbf{x}^{(k)}_{A_i}|\mathbf{x}^{(k)}_{\backslash A_i}).$$

This is the objective function in *conditional composite likelihood* [24, 30, 23, 1] (also rediscovered under the name *piecewise learning* in [26]).

A special case of conditional composite likelihood is when $A_i = \backslash\{i\}$, the resulting marginalization operator, $\Phi^m_{\backslash\{i\}}$, is known as the i[th] *singleton marginalization* operator. With the $d$ different singleton marginalization operators, we can rewrite the objective function as $\text{KL}(p\|q) - \frac{1}{d}\sum_{i=1}^{d}\text{KL}\left(\Phi^m_{\backslash i}p\big\|\Phi^m_{\backslash i}q\right) = \frac{1}{d}\sum_{i=1}^{d}\int_{\mathcal{R}} p_i(x_i)\text{KL}\left(p_{i|\backslash i}(\mathbf{x}_i|\mathbf{x}_{\backslash i})\big\|q_{i|\backslash i}(\mathbf{x}_i|\mathbf{x}_{\backslash i})\right) dx_i$. Note that in this case, the average KL contraction is zero if and only if $p(\mathbf{x})$ and $q_\theta(\mathbf{x})$ agree on every singleton conditional distribution, i.e., $p_{i|\backslash i}(\mathbf{x}_i|\mathbf{x}_{\backslash i}) = q_{i|\backslash i}(\mathbf{x}_i|\mathbf{x}_{\backslash i})$ for all $i$ and $\mathbf{x}$. According the Brook's Lemma [4], the latter condition is sufficient for $p(\mathbf{x}) = q_\theta(\mathbf{x})$ (*a.e.*). Furthermore, when approximating the expectations with averages over samples from $p(\mathbf{x})$, we have

$$\operatorname*{argmin}_{\theta} \mathrm{KL}(p\|q) - \frac{1}{d}\sum_{i=1}^{d}\mathrm{KL}\left(\Phi_{\backslash\{i\}}^{m}p \middle\| \Phi_{\backslash\{i\}}^{m}q\right) \approx \operatorname*{argmax}_{\theta} \frac{1}{n}\sum_{k=1}^{n}\frac{1}{d}\sum_{i=1}^{d}\log q_{i|\backslash i}(\mathbf{x}_i^{(k)}|\mathbf{x}_{\backslash i}^{(k)}),$$

which is objective function in maximum *pseudo-likelihood* learning [3, 29].

### 4.7 Relation with Marginal Composite Likelihood

We now consider combining Type III MKC objective function, Eq.(9), with the KL contraction operator constructed from the marginal grafting operation. Specifically, with $m$ different KL contraction operators constructed from marginal grafting on index subsets $A_1, \cdots, A_m$, using Lemma 2, we can expand the corresponding Type III minimum KL contraction objective function as:

$$\operatorname*{argmin}_{\theta} \mathrm{KL}(p\|q) - \frac{1}{m}\sum_{i=1}^{m}\mathrm{KL}\left(\Phi_{p,A_i}^{g}\{p\} \middle\| \Phi_{p,A_i}^{g}\{q\}\right) = \operatorname*{argmin}_{\theta} \frac{1}{m}\sum_{i=1}^{m}\mathrm{KL}(p_{A_i}(\mathbf{x}_{A_i})\|q_{A_i}(\mathbf{x}_{A_i}))$$

$$= \operatorname*{argmax}_{\theta} \frac{1}{m}\sum_{i=1}^{m}\int_{\mathcal{R}^d} p_{A_i}(\mathbf{x}_{A_i})\log q_{A_i}(\mathbf{x}_{A_i})d\mathbf{x}_{A_i} \approx \operatorname*{argmax}_{\theta} \frac{1}{n}\sum_{k=1}^{n}\frac{1}{m}\sum_{i=1}^{m}\log q_{A_i}(\mathbf{x}_{A_i}^{(k)})$$

The last step, which maximizes the log likelihood of a set of marginal distributions on training data, corresponds to the objective function of *marginal composite likelihood* [30]. With $m = 1$, the resulting objective is used in the *maximum marginal likelihood* or *Type-II likelihood* learning [9].

## 5 Discussions

In this work, based on an information geometric view of parameter learning, we have described minimum KL contraction as a unifying principle for non-ML parameter learning, showing that the objective functions of several existing non-ML parameter learning methods can all be understood as instantiations of this principle with different KL contraction operators.

There are several directions that we would like to extend the current work. First, the proposed minimum KL contraction framework may be further generalized using the more general $f$-divergence [8], of which the KL divergence is a special case. With the more general framework, we hope to reveal further relations among other types of non-ML learning objectives [16, 25, 28, 27]. Second, in the current work, we have focused on the idealization of parametric learning as matching probability distributions. In practice, learning is most often performed on finite data set with an unknown underlying distribution. In such cases, asymptotic properties of the estimation as data volume increases, such as consistency, become essential. While many non-ML learning methods covered in this work have been shown to be consistent individually, the unification based on the minimum KL contraction may provide a general condition for such asymptotic properties. Last, understanding different existing non-ML learning objectives through minimizing KL contraction also provides a principled approach to devise new non-ML learning methods, by seeking new KL contraction operators, or new combinations of existing KL contraction operators.

**Acknowledgement** The author would like to thank Jascha Sohl-Dickstein, Michael DeWeese and Michael Gutmann for helpful discussions on an early version of this work. This work is supported by the *National Science Foundation* under the CAREER Award Grant No. 0953373.

## Footnotes

[1]Indeed, it is known that the KL divergence behaves like the squared Euclidean distance [6].

[2]We cite the original reference to this and subsequent results, which are recast using the terminology introduced in this work. Due to the limit of space, we defer formal proofs of these results to the supplementary materials.

## References

[1] Arthur U. Asuncion, Qiang Liu, Alexander T. Ihler, and Padhraic Smyth. Learning with blocks: Composite likelihood and contrastive divergence. In *AISTATS*, 2010. 2, 7

[2] L. Bahl, P. Brown, P. de Souza, and R. Mercer. Maximum mutual information estimation of hidden markov model parameters for speech recognition. In *ICASSP*, 1986. 1, 2, 7

[3] J. Besag. Statistical analysis of non-lattice data. *The Statistician*, 24:179–95, 1975. 1, 2, 7, 8

[4] D. Brook. On the distinction between the conditional probability and the joint probability approaches in the specification of nearest-neighbor systems. *Biometrika*, 3/4(51):481–483, 1964. 7

[5] M. Á. Carreira-Perpiñán and G. E. Hinton. On contrastive divergence learning. In *AISTATS*, 2005. 6

[6] T. Cover and J. Thomas. *Elements of Information Theory*. Wiley-Interscience, 2nd edition, 2006. 2, 3

[7] D. R. Cox. Partial likelihood. *Biometrika*, 62(2):pp. 269–276, 1975. 1, 2, 6

[8] I. Csiszár and P. C. Shields. Information theory and statistics: A tutorial. *Foundations and Trends in Communications and Information Theory*, 1(4):417–528, 2004. 4, 8

[9] I.J. Good. *The Estimation of Probabilities: An Essay on Modern Bayesian Methods*. MIT Press, 1965. 1, 2, 8

[10] M. Gutmann and J. Hirayama. Bregman divergence as general framework to estimate unnormalized statistical models. In *Conference on Uncertainty in Artificial Intelligence (UAI)*, Barcelona, Spain, 2011. 2

[11] M. Gutmann and A. Hyvärinen. Noise-contrastive estimation: A new estimation principle for unnormalized statistical models. In *AISTATS*, 2010. 1, 2, 6

[12] G. E. Hinton. Training products of experts by minimizing contrastive divergence. *Neural Computation*, 14:1771–1800, 2002. 1, 2, 6

[13] P. J. Huber. Projection pursuit. *The Annuals of Statistics*, 13(2):435–475, 1985. 3

[14] A. Hyvärinen. Estimation of non-normalized statistical models using score matching. *Journal of Machine Learning Research*, 6:695–709, 2005. 1, 2, 7

[15] A. Hyvärinen. Connections between score matching, contrastive divergence, and pseudolikelihood for continuous-valued variables. *IEEE Transactions on Neural Networks*, 18(5):1529–1531, 2007. 2

[16] A. Hyvärinen. Some extensions of score matching. *Computational Statistics & Data Analysis*, 51:2499–2512, 2007. 8

[17] T. Jebara and A. Pentland. Maximum conditional likelihood via bound maximization and the CEM algorithm. In *NIPS*, 1998. 1, 2, 7

[18] J. Laurie Kindermann, Ross; Snell. *Markov Random Fields and Their Applications*. American Mathematical Society, 1980. 1

[19] E. Kreyszig. *Introductory Functional Analysis with Applications*. Wiley, 1989. 2, 4

[20] Stefan L. Lauritzen. Statistical manifolds. In *Differential Geometry in Statistical Inference*, pages 163–216, 1987. 4

[21] Lucien Le Cam. Maximum likelihood — an introduction. *ISI Review*, 58(2):153–171, 1990. 1

[22] Y. LeCun, S. Chopra, R. Hadsell, M. Ranzato, and F. Huang. Tutorial on energy-based learning. In *Predicting Structured Data*. MIT Press, 2006. 2

[23] P. Liang and M. I Jordan. An asymptotic analysis of generative, discriminative, and pseudolikelihood estimators. In *International Conference on Machine Learning*, 2008. 7

[24] B. G Lindsay. Composite likelihood methods. *Contemporary Mathematics*, 80(1):22–39, 1988. 1, 2, 7

[25] S. Lyu. Interpretation and generalization of score matching. In *UAI*, 2009. 7, 8

[26] A. McCallum, C. Pal, G. Druck, and X. Wang. Multi-conditional learning: Generative/discriminative training for clustering and classification. In *Association for the Advancement of Artificial Intelligence (AAAI)*, 2006. 7

[27] M. Pihlaja, M. Gutmann, and A. Hyvärinen. A family of computationally efficient and simple estimators for unnormalized statistical models. In *UAI*, 2010. 8

[28] J. Sohl-Dickstein, P. Battaglino, and M. DeWeese. Minimum probability flow learning. In *ICML*, 2011. 8

[29] D. Strauss and M. Ikeda. Pseudolikelihood estimation for social networks. *Journal of the American Statistical Association*, 85:204–212, 1990. 8

[30] C. Varin and P. Vidoni. A note on composite likelihood inference and model selection. *Biometrika*, 92(3):519–528, 2005. 2, 7, 8

[31] D. Vickrey, C. Lin, and D. Koller. Non-local contrastive objectives. In *ICML*, 2010. 1, 2, 6

